# Fisher Scoring and a Mixture of Modes Approach for Approximate Inference and Learning in Nonlinear State Space Models

**Thomas Briegel and Volker Tresp**
Siemens AG, Corporate Technology
Dept. Information and Communications
Otto-Hahn-Ring 6, 81730 Munich, Germany
{Thomas.Briegel, Volker.Tresp}@mchp.siemens.de

## Abstract

We present Monte-Carlo generalized EM equations for learning in nonlinear state space models. The difficulties lie in the Monte-Carlo E-step which consists of sampling from the posterior distribution of the hidden variables given the observations. The new idea presented in this paper is to generate samples from a Gaussian approximation to the true posterior from which it is easy to obtain independent samples. The parameters of the Gaussian approximation are either derived from the extended Kalman filter or the Fisher scoring algorithm. In case the posterior density is multimodal we propose to approximate the posterior by a sum of Gaussians (mixture of modes approach). We show that sampling from the approximate posterior densities obtained by the above algorithms leads to better models than using point estimates for the hidden states. In our experiment, the Fisher scoring algorithm obtained a better approximation of the posterior mode than the EKF. For a multimodal distribution, the mixture of modes approach gave superior results.

## 1 INTRODUCTION

Nonlinear state space models (NSSM) are a general framework for representing nonlinear time series. In particular, any NARMAX model (nonlinear auto-regressive moving average model with external inputs) can be translated into an equivalent NSSM. Mathematically, a NSSM is described by the system equation

$$x_t = f_w(x_{t-1}, u_t) + \epsilon_t \tag{1}$$

where $x_t$ denotes a hidden state variable, $\epsilon_t$ denotes zero-mean uncorrelated Gaussian noise with covariance $Q_t$ and $u_t$ is an exogenous (deterministic) input vector. The time-series measurements $y_t$ are related to the unobserved hidden states $x_t$ through the observation equation

$$y_t = g_v(x_t, u_t) + v_t \tag{2}$$

where $v_t$ is uncorrelated Gaussian noise with covariance $V_t$. In the following we assume that the nonlinear mappings $f_w(.)$ and $g_v(.)$ are neural networks with weight vectors $w$ and $v$, respectively. The initial state $x_0$ is assumed to be Gaussian distributed with mean $a_0$ and covariance $Q_0$. All variables are in general multidimensional. The two challenges

in NSSMs are the interrelated tasks of inference and learning. In inference we try to estimate the states of unknown variables $x_s$ given some measurements $y_1, \ldots, y_t$ (typically the states of past $(s < t)$, present $(s = t)$ or future $(s > t)$ values of $x_t$) and in learning we want to adapt some unknown parameters in the model (i.e. neural network weight vectors $w$ and $v$) given a set of measurements.[1] In the special case of linear state space models with Gaussian noise, efficient algorithms for inference and maximum likelihood learning exist. The latter can be implemented using EM update equations in which the E-step is implemented using forward-backward Kalman filtering (Shumway & Stoffer, 1982). If the system is nonlinear, however, the problem of inference and learning leads to complex integrals which are usually considered intractable (Anderson & Moore, 1979). A useful approximation is presented in section 3 where we show how the learning equations for NSSMs can be implemented using two steps which are repeated until convergence. First in the (Monte-Carlo) E-step, random samples are generated from the unknown variables (e.g. the hidden variables $x_t$) given the measurements. In the second step (a generalized M-step) those samples are treated as real data and are used to adapt $f_w(.)$ and $g_v(.)$ using some version of the backpropagation algorithm. The problem lies in the first step, since it is difficult to generate independent samples from a general multidimensional distribution. Since it is difficult to generate samples from the *proper* distribution the next best thing might be to generate samples using an *approximation* to the proper distribution which is the idea pursued in this paper. The first thing which might come to mind is to approximate the posterior distribution of the hidden variables by a multidimensional Gaussian distribution since generating samples from such a distribution is simple. In the first approach we use the extended Kalman filter and smoother to obtain mode and covariance of this Gaussian.[2] Alternatively, we estimate the mode and the covariance of the posterior distribution using an efficient implementation of Fisher scoring derived by Fahrmeir and Kaufmann (1991) and use those as parameters of the Gaussian. In some cases the approximation of the posterior mode by a single Gaussian might be considered too crude. Therefore, as a third solution, we approximate the posterior distribution by a sum of Gaussians (mixture of modes approach). Modes and covariances of those Gaussians are obtained using the Fisher scoring algorithm. The weights of the Gaussians are derived from the likelihood of the observed data given the individual Gaussian. In the following section we derive the gradient of the log-likelihood with respect to the weights in $f_w(.)$ and $g_v(.)$. In section 3, we show that the network weights can be updated using a Monte-Carlo E-step and a generalized M-step. Furthermore, we derive the different Gaussian approximations to the posterior distribution and introduce the mixture of modes approach. In section 4 we validate our algorithms using a standard nonlinear stochastic time-series model. In section 5 we present conclusions.

## 2  THE GRADIENTS FOR NONLINEAR STATE SPACE MODELS

Given our assumptions we can write the joint probability of the complete data for $t = 1, \ldots, T$ as[3]

$$p(X_T, Y_T, U_T) = p(U_T)\, p(x_0) \prod_{t=1}^{T} p(x_t | x_{t-1}, u_t) \prod_{t=1}^{T} p(y_t | x_t, u_t) \qquad (3)$$

where $U_T = \{u_1, \ldots, u_T\}$ is a set of *known* inputs which means that $p(U_T)$ is irrelevant in the following. Since only $Y_T = \{y_1, \ldots, y_T\}$ and $U_T$ are observed, the log-likelihood of the model is

$$\log L = \log \int p(X_T, Y_T | U_T) p(U_T) \, dX_T \propto \log \int p(X_T, Y_T | U_T) \, dX_T \tag{4}$$

with $X_T = \{x_0, \ldots, x_T\}$. By inserting the Gaussian noise assumptions we obtain the gradients of the log-likelihood with respect to the neural network weight vectors $w$ and $v$, respectively (Tresp & Hofmann, 1995)

$$\frac{\partial \log L}{\partial w} \propto \sum_{t=1}^{T} \iint \frac{\partial f_w(x_{t-1}, u_t)}{\partial w} (x_t - f_w(x_{t-1}, u_t)) \, p(x_t, x_{t-1} | Y_T, U_T) \, dx_{t-1} \, dx_t$$

$$\frac{\partial \log L}{\partial v} \propto \sum_{t=1}^{T} \int \frac{\partial g_v(x_t, u_t)}{\partial v} (y_t - g_v(x_t, u_t)) \, p(x_t | Y_T, U_T) \, dx_t. \tag{5}$$

## 3 APPROXIMATIONS TO THE E-STEP

### 3.1 Monte-Carlo Generalized EM Learning

The integrals in the previous equations can be solved using Monte-Carlo integration which leads to the following learning algorithm.

1. Generate $S$ samples $\{\hat{x}_0^s, \ldots, \hat{x}_T^s\}_{s=1}^{S}$ from $p(X_T | Y_T, U_T)$ assuming the current model is correct (Monte-Carlo E-Step).

2. Treat those samples as real data and update $w^{\text{new}} = w^{\text{old}} + \eta \frac{\partial \log L}{\partial w}$ and $v^{\text{new}} = v^{\text{old}} + \eta \frac{\partial \log L}{\partial v}$ with stepsize $\eta$ and

$$\frac{\partial \log L}{\partial w} \propto \frac{1}{S} \sum_{t=1}^{T} \sum_{s=1}^{S} \frac{\partial f_w(x_{t-1}, u_t)}{\partial w} \bigg|_{x_{t-1} = \hat{x}_{t-1}^s} (\hat{x}_t^s - f_w(\hat{x}_{t-1}^s, u_t)) \tag{6}$$

$$\frac{\partial \log L}{\partial v} \propto \frac{1}{S} \sum_{t=1}^{T} \sum_{s=1}^{S} \frac{\partial g_v(x_t, u_t)}{\partial v} \bigg|_{x_t = \hat{x}_t^s} (y_t - g_v(\hat{x}_t^s, u_t)) \tag{7}$$

(generalized M-step). Go back to step one.

The second step is simply a stochastic gradient step. The computational difficulties lie in the first step. Methods which produce samples from multivariate distributions such as Gibbs sampling and other Markov chain Monte-Carlo methods have (at least) two problems. First, the sampling process has to "forget" its initial condition which means that the first samples have to be discarded and there are no simple analytical tools available to determine how many samples must be discarded. Secondly, subsequent samples are highly correlated which means that many samples have to be generated before a sufficient amount of independent samples is available. Since it is so difficult to sample from the correct posterior distribution $p(X_T | Y_T, U_T)$ the idea in this paper is to generate samples from an approximate distribution from which it is easy to draw samples. In the next sections we present approximations using a multivariate Gaussian and a mixture of Gaussians.

### 3.2 Approximate Mode Estimation Using the Extended Kalman Filter

Whereas the Kalman filter is an optimal state estimator for linear state space models the extended Kalman filter is a suboptimal state estimator for NSSMs based on local linearizations of the nonlinearities.[4] The *extended Kalman filter and smoother* (EKFS) algorithm is

a forward-backward algorithm and can be derived as an approximation to posterior mode estimation for Gaussian error sequences (Sage & Melsa, 1971). Its application to our framework amounts to approximating $x_t^{\text{mode}} \approx \hat{x}_t^{\text{EKFS}}$ where $\hat{x}_t^{\text{EKFS}}$ is the smoothed estimate of $x_t$ obtained from forward-backward extended Kalman filtering over the set of measurements $Y_T$ and $x_t^{\text{mode}}$ is the mode of the posterior distribution $p(x_t|Y_T, U_T)$. We use $\hat{x}_t^{\text{EKFS}}$ as the center of the approximating Gaussian. The EKFS also provides an estimate of the error covariance of the state vector at each time step $t$ which can be used to form the covariance matrix of the approximating Gaussian. The EKFS equations can be found in Anderson & Moore (1979). To generate samples we recursively apply the following algorithm. Given $\hat{x}_{t-1}^s$ is a sample from the Gaussian approximation of $p(x_{t-1}|Y_T, U_T)$ at time $t-1$ draw a sample $\hat{x}_t^s$ from $p(x_t|x_{t-1} = \hat{x}_{t-1}^s, Y_T, U_T)$. The last conditional density is Gaussian with mean and covariance calculated from the EKFS approximation and the lag-one error covariances derived in Shumway & Stoffer (1982), respectively.

### 3.3 Exact Mode Estimation Using the Fisher Scoring Algorithm

If the system is highly nonlinear, however, the EKFS can perform badly in finding the posterior mode due to the fact that it uses a first order Taylor series expansion of the nonlinearities $f_w(.)$ and $g_v(.)$ (for an illustration, see Figure 1). A useful – and computationally tractable – alternative to the EKFS is to compute the "exact" posterior mode by maximizing $\log p(X_T|Y_T, U_T)$ with respect to $X_T$. A suitable way to determine a stationary point of the log posterior, or equivalently, of $p(X_T, Y_T|U_T)$ (derived from (3) by dropping $p(U_T)$) is to apply *Fisher scoring*. With the current estimate $X_T^{\text{FS,old}}$ we get a better estimate $X_T^{\text{FS,new}} = X_T^{\text{FS,old}} + \eta\,\delta$ for the unknown state sequence $X_T$ where $\delta$ is the solution of

$$\mathcal{S}(X_T^{\text{FS,old}})\,\delta = s(X_T^{\text{FS,old}}) \tag{8}$$

with the score function $s(X_T) = \frac{\partial \log p(X_T, Y_T|U_T)}{\partial X_T}$ and the expected information matrix $\mathcal{S}(X_T) = \mathrm{E}\left[-\frac{\partial^2 \log p(X_T, Y_T|U_T)}{\partial X_T \partial X_T^T}\right]$.[5] By extending the arguments given in Fahrmeir & Kaufmann (1991) to nonlinear state space models it turns out that solving equation (8) – e.g. to compute the inverse of the expected information matrix – can be performed by Cholesky decomposition in one forward and backward pass.[6] The forward-backward steps can be implemented as a fast EKFS-like algorithm which has to be iterated to obtain the maximum posterior estimates $x_t^{\text{mode}} = \hat{x}_t^{\text{FS}}$ (see Appendix). Figure 1 shows the estimate obtained by the Fisher scoring procedure for a bimodal posterior density. Fisher scoring is successful in finding the "exact" mode, the EKFS algorithm is not. Samples of the approximating Gaussian are generated in the same way as in the last section.

### 3.4 The Mixture of Modes Approach

The previous two approaches to posterior mode smoothing can be viewed as single Gaussian approximations of the mode of $p(X_T|Y_T, U_T)$. In some cases the approximation of the posterior density by a single Gaussian might be considered too crude, in particular if the posterior distribution is multimodal. In this section we approximate the posterior by a *weighted sum* of $m$ Gaussians $p(X_T|Y_T, U_T) \approx \sum_{k=1}^m \alpha^k p(X_T|k)$ where $p(X_T|k)$ is the $k$-th Gaussian. If the individual Gaussians model the different modes we are able to model multimodal posterior distributions accurately. The approximations of the individual modes are local maxima of the Fisher scoring algorithm which are found by starting the algorithm using different initial conditions. Given the different Gaussians, the optimal weighting factors are $\alpha^k = p(Y_T|k)p(k)/p(Y_T)$ where $p(Y_T|k) = \int p(Y_T|X_T)p(X_T|k)\,dX_T$ is the

likelihood of the data given mode $k$. If we approximate that integral by inserting the Fisher scoring solutions $\hat{x}_t^{\text{FS},k}$ for each time step $t$ and linearize the nonlinearity $g_v(.)$ about the Fisher scoring solutions, we obtain a closed form solution for computing the $\alpha^k$ (see Appendix). The resulting estimator is a weighted sum of the $m$ single Fisher scoring estimates $\hat{x}_t^{\text{MM}} = \sum_{k=1}^{m} \alpha^k \hat{x}_t^{\text{FS},k}$. The mixture of modes algorithm can be found in the Appendix. For the learning task samples of the mixture of Gaussians are based on samples of each of the $m$ single Gaussians which are obtained the same way as in subsection 3.2.

## 4 EXPERIMENTAL RESULTS

In the first experiment we want to test how well the different approaches can approximate the posterior distribution of a nonlinear time series (inference). As a time-series model we chose

$$f(x_{t-1}, u_t) = 0.5 x_{t-1} + 25 \frac{x_{t-1}}{1 + x_{t-1}^2} + 8 \cos(1.2(t-1)), \quad g(x_t) = \frac{1}{20} x_t^2, \quad (9)$$

the covariances $Q_t = 10$, $V_t = 1$ and initial conditions $a_0 = 0$ and $Q_0 = 5$ which is considered a hard inference problem (Kitagawa, 1987). At each time step we calculate the expected value of the hidden variables $x_t, t = 1, ..., 400$ based on a set of measurements $Y_{400} = \{y_1, ..., y_{400}\}$ (which is the optimal estimator in the mean squared sense) and based on the different approximations presented in the last section. Note that for the single mode approximation, $x_t^{\text{mode}}$ is the best estimate of $x_t$ based on the approximating Gaussian. For the mixture of modes approach, the best estimate is $\sum_{k=1}^{m} \alpha^k \hat{x}_t^{\text{FS},k}$ where $\hat{x}_t^{\text{FS},k}$ is the mode of the $k$-th Gaussian in the dimension of $x_t$. Figure 2 (left) shows the mean squared error (MSE) of the smoothed estimates using the different approaches. The Fisher scoring (FS) algorithm is significantly better than the EKFS approach. In this experiment, the mixture of modes (MM) approach is significantly better than both the EKFS and Fisher scoring. The reason is that the posterior probability is multimodal as shown in Figure 1.

In the second experiment we used the same time-series model and trained a neural network to approximate $f_w(.)$ where all covariances were assumed to be fixed and known. For adaptation we used the learning rules of section 3 using the various approximations to the posterior distribution of $X_T$. Figure 2 (right) shows the results. The experiments show that truly sampling from the approximating Gaussians gives significantly better results than using the expected value as a point estimate. Furthermore, using the mixture of modes approach in conjunction with sampling gave significantly better results than the approximations using a single Gaussian. When used for inference, the network trained using the mixture of modes approach was not significantly worse than the true model (5% significance level, based on 20 experiments).

## 5 CONCLUSIONS

In our paper we presented novel approaches for inference and learning in NSSMs. The application of Fisher scoring and the mixture of modes approach to nonlinear models as presented in our paper is new. Also the idea of sampling from an approximation to the posterior distribution of the hidden variables is presented here for the first time. Our results indicate that the Fisher scoring algorithm gives better estimates of the expected value of the hidden variable than the EKFS based approximations. Note that the Fisher scoring algorithm is more complex in requiring typically 5 forward-backward passes instead of only one forward-backward pass for the EKFS approach. Our experiments also showed that if the posterior distribution is multimodal, the mixture of modes approach gives significantly better estimates if compared to the approaches based on a single Gaussian approximation. Our learning experiments show that it is important to sample from the approximate distributions and that it is not sufficient to simply substitute point estimates. Based on the

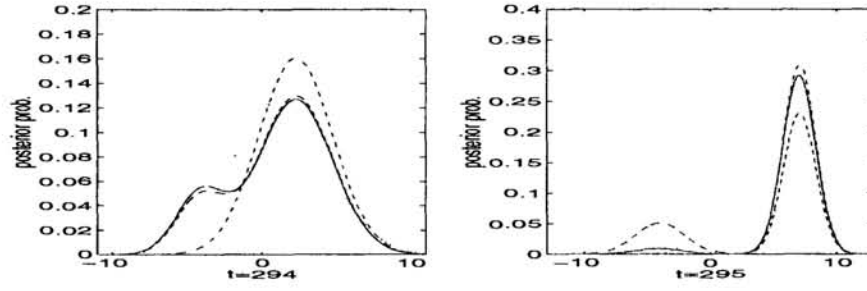

Figure 1: Approximations to the posterior distribution $p(x_t|Y_{400}, U_{400})$ for $t = 294$ and $t = 295$. The continuous line shows the posterior distribution based on Gibbs sampling using 1000 samples and can be considered a close approximation to the true posterior. The EKFS approximation (dotted) does not converge to a mode. The Fisher scoring solution (dash-dotted) finds the largest mode. The mixture of modes approach with 50 modes (dashed) correctly finds the two modes.

sampling approach it is also possible to estimate hyperparameters (e.g. the covariance matrices) which was not done in this paper. The approaches can also be extended towards online learning and estimation in various ways (e.g. missing data problems).

**Appendix: Mixture of Modes Algorithm**

The *mixture of modes* estimate $\hat{x}_t^{\mathrm{MM}}$ is derived as a weighted sum of $k = 1, \ldots, m$ individual Fisher scoring (mode) estimates $\hat{x}_t^{\mathrm{FS},k}$. For $m = 1$ we obtain the *Fisher scoring* algorithm of subsection 3.3.

First, one performs the set of forward recursions $(t = 1, \ldots, T)$ for each single mode estimator $k$.

$$\Sigma_{t|t-1}^k = F_t(\hat{x}_{t-1}^{\mathrm{FS},k})\Sigma_{t-1|t-1}^k F_t^{\mathsf{T}}(\hat{x}_{t-1}^{\mathrm{FS},k}) + Q_t \tag{10}$$

$$B_t^k = \Sigma_{t-1|t-1}^k F_t^{\mathsf{T}}(\hat{x}_{t-1}^{\mathrm{FS},k})(\Sigma_{t|t-1}^k)^{-1} \tag{11}$$

$$\Sigma_{t|t}^k = \left((\Sigma_{t|t-1}^k)^{-1} + G_t(\hat{x}_t^{\mathrm{FS},k})V_t^{-1}G_t^{\mathsf{T}}(\hat{x}_t^{\mathrm{FS},k})\right)^{-1} \tag{12}$$

$$\gamma_t^k = s_t(\hat{x}_t^{\mathrm{FS},k}) + B_t^{k\mathsf{T}}\gamma_{t-1}^k \tag{13}$$

with the initialization $\Sigma_{0|0}^k = Q_0$, $\gamma_0 = s_0(\hat{x}_0^{\mathrm{FS},k})$. Then, one performs the set of backward smoothing recursions $(t = T, \ldots, 1)$

$$(D_{t-1}^k)^{-1} = \Sigma_{t-1|t-1}^k - B_t^k \Sigma_{t|t-1}^k B_t^{k\mathsf{T}} \tag{14}$$

$$\Sigma_{t-1}^k = (D_{t-1}^k)^{-1} + B_t^k \Sigma_t^k B_t^{k\mathsf{T}} \tag{15}$$

$$\delta_{t-1}^k = (D_{t-1}^k)^{-1}\gamma_{t-1}^k + B_t^k \delta_t^k \tag{16}$$

with $F_t(z) = \frac{\partial f_w(x_{t-1}, u_t)}{\partial x_{t-1}}|_{x_{t-1}=z}$, $G_t(z) = \frac{\partial g_v(x_t, u_t)}{\partial x_t}|_{x_t=z}$, $s_t(z) = \frac{\partial \log p(X_T, Y_T|U_T)}{\partial x_t}|_{x_t=z}$ and initialization $\delta_T^k = \Sigma_T^k \gamma_T^k$. The $k$ individual mode estimates $\hat{x}_t^{\mathrm{FS},k}$ are obtained by iterative application of the update rule $X_T^{\mathrm{FS},k} := \eta \delta^k + X_T^{\mathrm{FS},k}$ with stepsize $\eta$ where $X_T^{\mathrm{FS},k} = \{\hat{x}_0^{\mathrm{FS},k}, \ldots, \hat{x}_T^{\mathrm{FS},k}\}$ and $\delta^k = \{\delta_0^k, \ldots, \delta_T^k\}$. After convergence we obtain the mixture of modes estimate as the weighted sum $\hat{x}_t^{\mathrm{MM}} = \sum_{k=1}^m \alpha^k \hat{x}_t^{\mathrm{FS},k}$ with weighting coefficients $\alpha^k := \alpha_0^k$ where $\alpha_t^k (t = T - 1, \ldots, 0)$ are computed recursively starting with a uniform prior $\alpha_T^k = \frac{1}{m}$ ($\mathcal{N}(x|\mu, \Sigma)$ stands for a Gaussian with center $\mu$ and covariance $\Sigma$ evaluated at $x$):

$$\alpha_t^k = \frac{\alpha_{t+1}^k \mathcal{N}(y_t|g_v(\hat{x}_t^{\mathrm{FS},k}, u_t), \Omega_t^k)}{\sum_{j=1}^m \alpha_{t+1}^j \mathcal{N}(y_t|g_v(\hat{x}_t^{\mathrm{FS},j}, u_t), \Omega_t^j)} \tag{17}$$

$$\Omega_t^k = G_t(\hat{x}_t^{\mathrm{FS},k})\Sigma_t^k G_t(\hat{x}_t^{\mathrm{FS},k})^{\mathsf{T}} + V_t \tag{18}$$

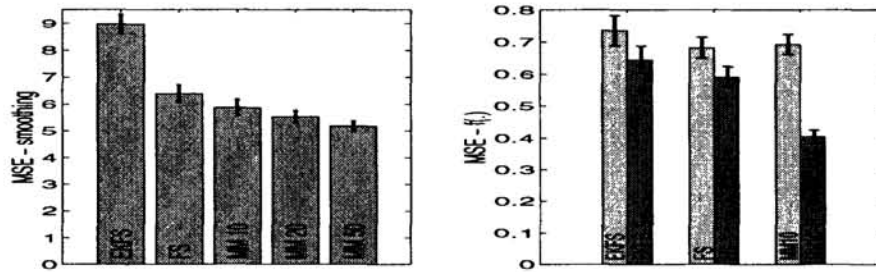

Figure 2: Left (inference): The heights of the bars indicate the mean squared error between the true $x_t$ (which we know since we simulated the system) and the estimates using the various approximations. The error bars show the standard deviation derived from 20 repetitions of the experiment. Based on the paired $t$-test, Fisher scoring is significantly better than the EKFS and all mixture of modes approaches are significantly better than both EKFS and Fisher scoring based on a 1% rejection region. The mixture of modes approximation with 50 modes (MM 50) is significantly better than the approximation using 20 modes. The improvement of the approximation using 20 modes (MM 20) is not significantly better than the approximation with 10 (MM 10) modes using a 5% rejection region.

Right (learning): The heights of the bars indicate the mean squared error between the true $f_w(.)$ (which is known) and the approximations using a multi-layer perceptron with 3 hidden units and $T = 200$. Shown are results using the EKFS approximation, (left) the Fisher scoring approximation (center) and the mixture of modes approximation (right). There are two bars for each experiment: The left bars show results where the expected value of $x_t$ calculated using the approximating Gaussians are used as (single) samples for the generalized M-step – in other words – we use a point estimate for $x_t$. Using the point estimates, the results of all three approximations are not significantly different based on a 5% significance level. The right bars shows the result where $S = 50$ samples are generated for approximating the gradient using the Gaussian approximations. The results using sampling are all significantly better than the results using point estimates (1% significance level). The sampling approach using the mixture of modes approximation is significantly better than the other two sampling-based approaches (1% significance level). If compared to the inference results of the experiments shown on the left, we achieved a mean squared error of 6.02 for the mixture of modes approach with 10 modes which is not significantly worse than the results the with the true model of 5.87 (5% significance level).

## Footnotes

[1] In this paper we focus on the case $s \leq t$ (smoothing and offline learning, respectively).

[2] Independently from our work, a single Gaussian approximation to the E-step using the EKFS has been proposed by Ghahramani & Roweis (1998) for the special case of a RBF network. They show that one obtains a closed form M-step when just adapting the *linear* parameters by holding the nonlinear parameters fixed. Although avoiding sampling, the computational load of their M-step seems to be significant.

[3] In the following, each probability density is conditioned on the current model. For notational convenience, we do not indicate this fact explicitly.

[4] Note that we do not include the parameters in the NSSM as additional states to be estimated as done by other authors, e.g. Puskorius & Feldkamp (1994).

[5]Note that the difference between the Fisher scoring and the Gauss-Newton update is that in the former we take the expectation of the information matrix.

[6]The expected information matrix is a positive definite block-tridiagonal matrix.

## References

Anderson, B. and Moore, J. (1979) *Optimal Filtering*, Prentice-Hall, New Jersey.

Fahrmeir, L. and Kaufmann, H. (1991) *On Kalman Filtering, Posterior Mode Estimation and Fisher Scoring in Dynamic Exponential Family Regression*, Metrika, 38, pp. 37-60.

Ghahramani, Z. and Roweis, S. (1999) *Learning Nonlinear Stochastic Dynamics using the Generalized EM Algorithm*, Advances in Neural Information Processing Systems 11, eds. M. Kearns, S. Solla, D. Cohn, MIT Press, Cambridge, MA.

Kitagawa, G. (1987) *Non-Gaussian State Space Modeling of Nonstationary Time Series (with Comments)*, JASA 82, pp. 1032-1063.

Puskorius, G. and Feldkamp, L. (1994) *Neurocontrol of Nonlinear Dynamical Systems with Kalman Filter Trained Recurrent Networks*, IEEE Transactions on Neural Networks, 5:2, pp. 279-297.

Sage, A. and Melsa, J. (1971) *Estimation Theory with Applications to Communications and Control*, McGraw-Hill, New York.

Shumway, R. and Stoffer, D. (1982) *Time Series Smoothing and Forecasting Using the EM Algorithm*, Technical Report No. 27, Division of Statistics, UC Davis.

Tresp, V. and Hofmann, R. (1995) *Missing and Noisy Data in Nonlinear Time-Series Prediction*, Neural Networks for Signal Processing 5, IEEE Sig. Proc. Soc., pp. 1-10.
